# Real-time Particle Filters

**Cody Kwok**[†]          **Dieter Fox**[†]          **Marina Meilă**[‡]

[†]Dept. of Computer Science & Engineering, [‡]Dept. of Statistics
University of Washington
Seattle, WA 98195
{ctkwok,fox}@cs.washington.edu, mmp@stat.washington.edu

## Abstract

Particle filters estimate the state of dynamical systems from sensor information. In many real time applications of particle filters, however, sensor information arrives at a significantly higher rate than the update rate of the filter. The prevalent approach to dealing with such situations is to update the particle filter as often as possible and to discard sensor information that cannot be processed in time. In this paper we present real-time particle filters, which make use of *all* sensor information even when the filter update rate is below the update rate of the sensors. This is achieved by representing posteriors as mixtures of sample sets, where each mixture component integrates one observation arriving during a filter update. The weights of the mixture components are set so as to minimize the approximation error introduced by the mixture representation. Thereby, our approach focuses computational resources (samples) on valuable sensor information. Experiments using data collected with a mobile robot show that our approach yields strong improvements over other approaches.

## 1   Introduction

Due to their sample-based representation, particle filters are well suited to estimate the state of non-linear dynamic systems. Over the last years, particle filters have been applied with great success to a variety of state estimation problems including visual tracking, speech recognition, and mobile robotics [1]. The increased representational power of particle filters, however, comes at the cost of higher computational complexity.

The application of particle filters to online, real-time estimation raises new research questions. The key question in this context is: *How can we deal with situations in which the rate of incoming sensor data is higher than the update rate of the particle filter?* To the best of our knowledge, this problem has not been addressed in the literature so far. The prevalent approach in real time applications is to update the filter as often as possible and to discard sensor information that arrives during the update process. Obviously, this approach is prone to losing valuable sensor information. At first sight, the sample based representation of particle filters suggests an alternative approach similar to an any-time implementation: Whenever a new observation arrives, sampling is interrupted and the next observation is processed. Unfortunately, such an approach can result in too small sample sets, causing the filter to diverge [1, 2].

In this paper we introduce real-time particle filters (RTPF) to deal with constraints imposed by limited computational resources. Instead of discarding sensor readings, we distribute the

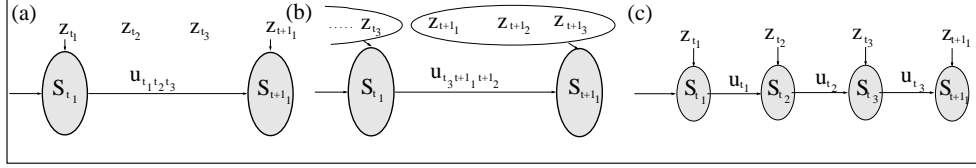

Figure 1: Different strategies for dealing with limited computational power. All approaches process the same number of samples per estimation interval (window sizeq three). (a) Skip observations, *i.e.* integrate only every third observation. (b) Aggregate observations within a window and integrate them in one step. (c) Reduce sample set size so that each observation can be considered.

samples among the different observations arriving during a filter update. Hence RTPF represents densities over the state space by *mixtures* of sample sets, thereby avoiding the problem of filter divergence due to an insufficient number of independent samples. The weights of the mixture components are computed so as to minimize the approximation error introduced by the mixture representation. The resuling approach naturally focuses computational resources (samples) on valuable sensor information.

The remainder of this paper is organized as follows: In the next section we outline the basics of particle filters in the context of real-time constraints. Then, in Section 3, we introduce our novel technique to real-time particle filters. Finally, we present experimental results followed by a discussion of the properties of RTPF.

## 2   Particle filters

Particle filters are a sample-based variant of Bayes filters, which recursively estimate posterior densities, or beliefs $Bel$, over the state $x_t$ of a dynamical system (see [1, 3] for details):

$$Bel(x_t) \quad \propto \quad p(z_t \mid x_t) \int p(x_t \mid x_{t-1}, u_{t-1}) \ Bel(x_{t-1}) \ dx_{t-1}. \tag{1}$$

Here $z_t$ is a sensor measurement and $u_{t-1}$ is control information measuring the dynamics of the system. Particle filters represent beliefs by sets $S_t$ of weighted samples $\langle x_t^{(i)}, w_t^{(i)} \rangle$. Each $x_t^{(i)}$ is a state, and the $w_t^{(i)}$ are non-negative numerical factors called *importance weights*, which sum up to one. The basic form of the particle filter realizes the recursive Bayes filter according to a sampling procedure, often referred to as sequential importance sampling with resampling (SISR):

1. *Resampling:* Draw with replacement a random state $x$ from the set $S_{t-1}$ according to the (discrete) distribution defined through the importance weights $w_{t-1}^{(i)}$.

2. *Sampling:* Use $x$ and the control information $u_{t-1}$ to sample $x'$ according to the distribution $p(x' \mid x, u_{t-1})$, which describes the dynamics of the system.

3. *Importance sampling:* Weight the sample $x'$ by the observation likelihood $w' = p(z_t \mid x')$.

Each iteration of these three steps generates a sample $\langle x', w' \rangle$ representing the posterior. After $n$ iterations, the importance weights of the samples are normalized so that they sum up to one. Particle filters can be shown to converge to the true posterior even in non-Gaussian, non-linear dynamic systems [4].

A typical assumption underlying particle filters is that all samples can be updated whenever new sensor information arrives. Under realtime conditions, however, it is possible that the update cannot be completed before the next sensor measurement arrives. This can be the case for computationally complex sensor models or whenever the underlying posterior requires large sample sets [2]. The majority of filtering approaches deals with this problem by skipping sensor information that arrives during the update of the filter. While this approach works reasonably well in many situations, it is prone to miss valuable sensor information.

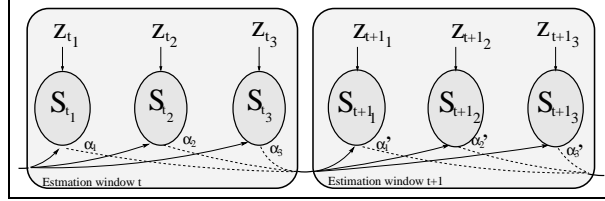

Figure 2: Real time particle filters. The $n$ samples are distributed among the observations within one estimation interval (window size three in this example). The belief is a mixture of the individual sample sets. Each arrow additionally represents the system dynamics $p(x_{t+1_i} \mid x_{t_j}, u_{t_{ij}})$.

Before we discuss ways of dealing with such situations, let us introduce some notation. We assume that observations arrive at time intervals $\Delta$, which we will call *observation intervals*. Let $n$ be the number of samples required by the particle filter. Assume that the resulting update cycle of the particle filter takes $k\Delta$ and is called the *estimation interval* or *estimation window*. Accordingly, $k$ observations arrive during one estimation interval. We call this number the *window size* of the filter, *i.e.* the number of observations obtained during a filter update. The $i$-th observation and state within window $t$ are denoted $z_{t_i}$ and $x_{t_i}$, respectively.

Fig. 1 illustrates different approaches to dealing with window sizes larger than one. The simplest and most common aproach is shown in Fig. 1(a). Here, observations arriving during the update of the sample set are discarded, which has the obvious disadvantage that valuable sensor information might get lost. The approach in Fig. 1(b) overcomes this problem by aggregating multiple observations into one. While this technique avoids the loss of information, it is not applicable to arbitrary dynamical systems. For example, it assumes that observations can be aggregated optimally, and that the integration of an aggregated observation can be performed as efficiently as the integration of individual observations, which is often not the case. The third approach, shown in Fig. 1(c), simply stops generating new samples whenever an observation is made (hence each sample set contains only $n/k$ samples). While this approach takes advantage of the any-time capabilities of particle filters, it is susceptible to filter divergence due to an insuffient number of samples [2, 1].

## 3 Real time particle filters

In this paper we propose real time particle filters (RTPFs), a novel approach to dealing with limited computational resources. The key idea of RTPFs is to consider *all* sensor measurements by distributing the samples among the observations within an update window. Additionally, by weighting the different sample sets within a window, our approach focuses the computational resources (samples) on the most valuable observations. Fig. 2 illustrates the approach. As can be seen, instead of one sample set at time $t$, we maintain $k$ smaller sample sets at $t_1, \ldots t_k$. We treat such a "virtual sample set", or belief, as a mixture of the distributions represented in it. The mixture components represent the state of the system at different points in time. If needed, however, the complete belief can be generated by considering the dynamics between the individual mixture components.

Compared to the first approach discussed in the previous section, this method has the advantage of not skipping any observations. In contrast to the approach shown in Fig. 1(b), RTPFs do not make any assumptions about the nature of the sensor data, *i.e.* whether it can be aggregated or not. The difference to the third approach (Fig. 1(c)) is more subtle. In both approaches, each of the $k$ sample sets can only contain $n/k$ samples. The belief state that is propagated by RTPF to the next estimation interval is a mixture distribution where each mixture component is represented by one of the $k$ sample sets, all generated independently from the previous window. Thus, the belief state propagation is simulated by $k \cdot \frac{n}{k}$ sample trajectories, that for computational convenience are represented at the points in time where the observations are integrated. In the approach (c) however, the belief propagation is simulated with only $n/k$ independent samples.

We will now show how RTPF determines the weights of the mixture belief. The key idea is to choose the weights that minimize the KL-divergence between the mixture belief and the optimal belief. The optimal belief is the belief we would get if there was enough time to compute the full posterior within the update window.

## 3.1 Mixture representation

Let us restrict our attention to one estimation interval consisting of $k$ observations. The optimal belief $Bel_{opt}(x_{t_k})$ at the end of an estimation window results from iterative application of the Bayes filter update on each obseration [3]:

$$Bel_{opt}(x_{t_k}) \propto \int \ldots \int \prod_{i=1}^{k} p(z_{t_i} \mid x_{t_i}) \, p(x_{t_i} \mid x_{t_{i-1}}, u_{t_{i-1}}) \, Bel(x_{t_0}) \, dx_{t_0} \ldots dx_{t_{k-1}}. \quad (2)$$

Here $Bel(x_{t_0})$ denotes the belief generated in the previous estimation window. In essence, (2) computes the belief by integrating over all *trajectories* through the estimation interval, where the start position of the trajectories is drawn from the previous belief $Bel(x_{t_0})$. The probability of each trajectory is determined using the control information $u_{t_0}, u_{t_1}, \ldots, u_{t_{k-1}}$, and the likelihoods of the observations $z_{t_1}, \ldots, z_{t_k}$ along the trajectory. Now let $Bel_i(x_{t_k})$ denote the belief resulting from integrating only the $i-th$ observation within the estimation window. RTPF computes a mixture of $k$ such beliefs, one for each observation. The mixture, denoted $Bel_{mix}(x_{t_k} \mid \alpha)$, is the weighted sum of the mixture components $Bel_i(x_{t_k})$, where $\alpha$ denotes the mixture weights:

$$Bel_{mix}(x_{t_k} \mid \alpha) = \sum_{i=1}^{k} \alpha_i Bel_i(x_{t_k})$$

$$\propto \sum_{i=1}^{k} \alpha_i \int \ldots \int p(z_{t_i} \mid x_{t_i}) \prod_{j=1}^{k} p(x_{t_j} \mid x_{t_{j-1}}, u_{t_{j-1}}) Bel(x_{t_0}) dx_{t_0} \ldots dx_{t_{k-1}}. \quad (3)$$

where $\alpha_i \geq 0$ and $\sum_i \alpha_i = 1$. Here, too, we integrate over all trajectories. In contrast to (2), however, each trajectory selectively integrates only one of the $k$ observations within the estimation interval[1].

## 3.2 Optimizing the mixture weights

We will now turn to the problem of finding the weights of the mixture. These weights reflect the "importance" of the respective observations for describing the optimal belief. The idea is to set them so as to minimize the approximation error introduced by the mixture distribution. More formally, we determine the mixing weights $\alpha^*$ by minimizing the KL-divergence [5] between $Bel_{mix}$ and $Bel_{opt}$.

$$\alpha^* = \operatorname*{argmin}_{\alpha \in \mathcal{A}} KL(Bel_{mix}(\cdot \mid \alpha) \| Bel_{opt}) \quad (4)$$

$$= \operatorname*{argmin}_{\alpha \in \mathcal{A}} \int Bel_{mix}(x_{t_k} \mid \alpha) \, log \, \frac{Bel_{mix}(x_{t_k} \mid \alpha)}{Bel_{opt}(x_{t_k})} \, dx_{t_k}. \quad (5)$$

In the above $\mathcal{A} = \{\alpha \mid \sum_{i=1}^{k} \alpha_i = 1, \ \alpha_i \geq 0\}$. Optimizing the weights of mixture approximations can be done using EM [6] or (constrained) gradient descent [7]. Here, we perform a small number of gradient descent steps to find the mixture weights. Denote by

$J(\alpha)$ the criterion to be minimized in (5). The gradient of $J(\alpha)$ is given by

$$\frac{\partial J}{\partial \alpha_i} = \int \frac{\partial}{\partial \alpha_i} Bel_{mix}(x_{t_k}|\alpha) \log Bel_{mix}(x_{t_k}|\alpha) - \frac{\partial}{\partial \alpha_i} Bel_{mix}(x_{t_k}|\alpha) \log Bel_{opt}(x_{t_k})$$

$$= 1 + \int Bel_i(x_{t_k}) \log \frac{Bel_{mix}(x_{t_k}|\alpha)}{Bel_{opt}(x_{t_k})} dx_{t_k}, \qquad i = 1, \dots k. \tag{6}$$

The start point $\alpha_s$ for the gradient descent is chosen to be the center of the weight domain $\mathcal{A}$, that is $\alpha_s = [\frac{1}{k} \ \frac{1}{k} \ \dots \ \frac{1}{k}]^T$.

### 3.3  Monte Carlo gradient estimation

The exact computation of the gradients in (6) requires the computation of the different beliefs, each in turn requiring several particle filter updates (see (2), (3)), and integration over all states $x_{t_k}$. This is clearly not feasible in our case. We solve this problem by Monte Carlo approximation. The approach is based on the observation that the beliefs in (6) share the same trajectories through space and differ only in the observations they integrate. Therefore, we first generate sample trajectories through the estimation window *without* considering the observations, and then use importance sampling to generate the beliefs needed for the gradient estimation. Trajectory generation is done as follows: we draw a sample $x_{t-1}$ from a sample set of the previous mixture belief, where the probability of chosing a set $S_{t-1_j}$ is given by the mixture weights $\alpha_j$. This sample is then moved forward in time by consecutively drawing samples $x_{t_i}$ from the distributions $p(x_{t_i} \mid x_{t_{i-1}}, u_{t_{i-1}})$ at each time step $t_i$, $i = 1, \dots k$. The resulting trajectories are drawn from the following proposal distribution $f$:

$$f(x_{t_k}) = \int \dots \int \prod_{i=1}^{k} p(x_{t_i} \mid x_{t_{i-1}}, u_{t_{i-1}}) Bel(x_{t_0}) dx_{t_0} \dots dx_{t_{k-1}} \tag{7}$$

Using importance sampling, we obtain sample-based estimates of $Bel_i$ and $Bel_{opt}$ by simply weighting each trajectory with $p(z_{t_i} \mid x_{t_i})$ or $\prod_{j=1}^{k} p(z_{t_j} \mid x_{t_j})$, respectively (compare (2) and (3)). $Bel_{mix}$ is generated with minimal computational overhead by averaging the weights computed for the individual $Bel_i$ distributions. The use of the same trajectories for all distributions has the advantage that it is highly efficient and that it reduces the variance of the gradient estimate. This variance reduction is due to using the same random bits in evaluating the diverse scenarios of incorporating one or another of the observations [8].

Further variance reduction is achieved by using stratified sampling on trajectories. The trajectories are grouped by determining connected regions in a grid over the state space (at time $t_1$). Neighboring cells are considered connected if both contain samples. To compute the gradients by formula (6), we then perform summation and normalization over the grouped trajectories. Empirical evaluations showed that this grouping greatly reduces the number of trajectories needed to get smooth gradient estimates. An additional, very important benefit of grouping is the reduction of the bias due to different dynamics applied to the different sample sets in the estimation window. In our experiments the number of trajectories is less than 2% of the total number of samples, resulting in a computational overhead of about 1% of the total estimation time.

To summarize, the RTPF algorithm works as follows. The number $n$ of independent samples needed to represent the belief, the update rate of incoming sensor data, and the available processing power determine the size $k$ of the estimation window and hence the number of mixture components. RTPF computes the optimal weights of the mixture distribution at the end of each estimation window. This is done by gradient descent using the Monte Carlo estimates of the gradients. The resulting weights are used to generate samples for the individual sample sets of the next estimation window. To do so, we keep track of the control information (dynamics) between the different sample sets of two consecutive windows.

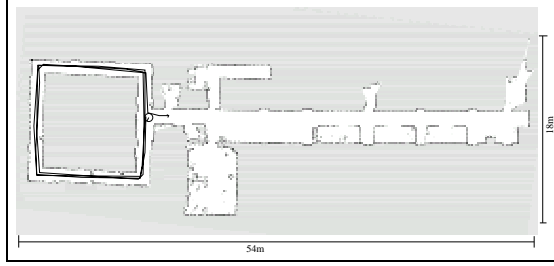

Fig. 3: Map of the environment used for the experiment. The robot was moved around the symmetric loop on the left. The task of the robot was to determine its position using data collected by two distance measuring devices, one pointing to its left, the other pointing to its right.

## 4   Experiments

In this section we evaluate the effectiveness of RTPF against the alternatives, using data collected from a mobile robot in a real-world environment. Figure 3 shows the setup of the experiment: The robot was placed in the office floor and moved around the loop on the left. The task of the robot was to determine its position within the map, using data collected by two laser-beams, one pointing to its left, the other pointing to its right. The two laser beams were extracted from a planar laser range-finder, allowing the robot only to determine the distance to the walls on its left and right. Between each observation the robot moved approximately 50cm (see [3] for details on robot localization and sensor models). Note that the loop in the environment is symmetric except for a few "landmarks" along the walls of the corridor. Localization performance was measured by the average distance between the samples and the reference robot positions, which were computed offline.

In the experiments, our real-time algorithm, RTPF, is compared to particle filters with skipping observations, called "Skip data" (Figure 1a), and particle filters with insufficient samples, called "Naive" (Figure 1c). Furthermore, to gauge the efficiency of our mixture weighting, we also obtained results for our real-time algorithm without weighting, *i.e.* we used mixture distributions and fixed the weights to $1/k$. We denote this variant "Uniform". Finally, we also include as reference the "Baseline" approach, which is allowed to generate $n$ samples for each observation, thereby not considering real-time constraints.

The experiment is set up as follows. First, we fix the sample set size $n$ which is sufficient for the robot to localize itself. In our experiment $n$ is set empirically to 20,000 (the particle filters may fail at lower $n$, see also [2]). We then vary the computational resources, resulting in different window sizes $k$. Larger window size means lower computational power, and the number of samples that can be generated for each observation decreases to $(n/k)$.

Figure 4 shows the evolutions of average localization errors over time, using different window sizes. Each graph is obtained by averaging over 30 runs with different random seeds and start positions. The error bars indicate 95% confidence intervals. As the figures show, "Naive" gives the worst results, which is due to insufficient numbers of samples, resulting in divergence of the filter. While "Uniform" performs slightly better than "Skip data", RTPF is the most effective of all algorithms, localizing the robot in the least amount of time. Furthermore, RTPF shows the least degradation with limited computational power (larger window sizes). The key advantage of RTPF over "Uniform" lies in the mixture weighting, which allows our approach to focus computational resources on valuable sensor information, for example when the robot passes an informative feature in one of the hallways. For short window sizes (Fig. 4(a)), this advantage is not very strong since in this environment, most features can be detected in several consecutive sensor measurements. Note that because the "Baseline" approach was allowed to integrate all observations with all of the 20,000 samples, it converges to a lower error level than all the other approaches.

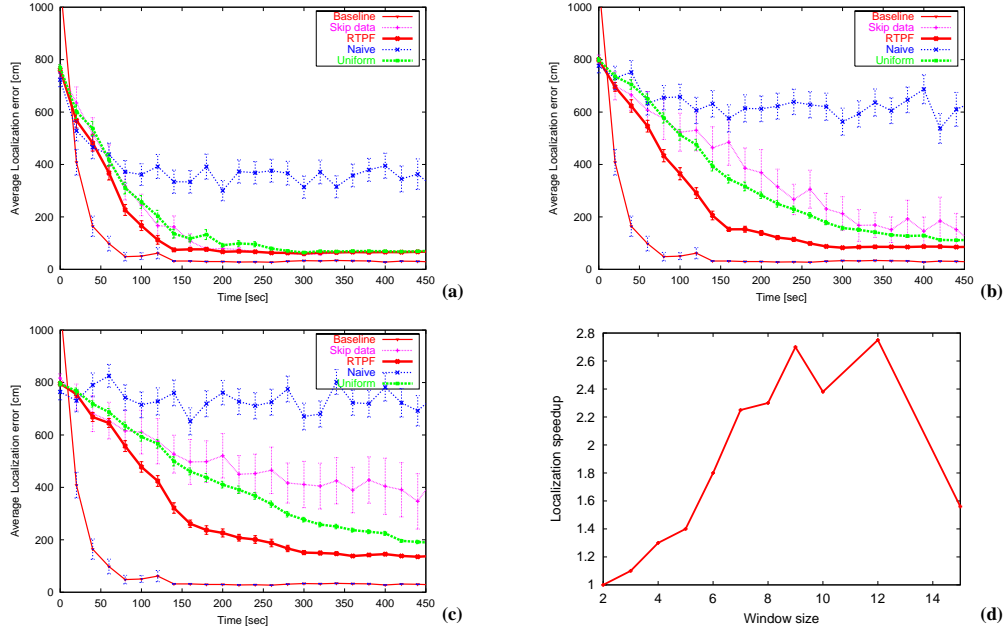

Fig. 4(a)-(c): Performance of the different algorithms for window sizes of 4, 8, and 12 respectively. The $x$-axis represents time elapsed since the beginning of the localization experiment. The $y$-axis plots the localization error measured in average distance from the reference position. Each figure includes the performance achieved with unlimited computational power as the "Baseline" graph. Each point is averaged over 30 runs, and error bars indicate 95% confidence intervals. Fig. 4(d) represents the localization speedup of RTPF over "Skip data" for various window sizes. The advantage of RTPF increases with the difficulty of the task, *i.e.* with increasing window size. Between window size 6 and 12, RTPF localizes at least twice as fast as "Skip data".

Without mixture weighting of RTPF, we did not expect "Uniform" to outperform "Skip data" significantly. To see this, consider one estimation window of length $k$. Suppose only one of the $k$ observations detects a landmark, or very informative feature in the hallway. In such a situation, "Uniform" considers this landmark every time the robot passes it. However, it only assigns $n/k$ samples to this landmark detection. "Skip data" on the other hand, detects the landmark only every $k$-th time, but assigns *all $n$* samples to it. Therefore, averaged over many different runs, the mean performance of "Uniform" and "Skip data" is very similar. However, the variance of the error is significantly lower for "Uniform" since it considers the detection in every run. In contrast to both approaches, RTPF detects all landmarks *and* generates more samples for the landmark detections, thereby gaining the best of both worlds, and Figures 4(a)–(c) show this is indeed the case.

In Figure 4(d) we summarize the performance gain of RTPF over "Skip data" for different window sizes in terms of localization time. We considered the robot to be localized if the average localization error remains below 200 cm over a period of 10 seconds. If the run never reaches this level, the localization time is set to the length of the entire run, which is 574 seconds. The $x$-axis represents the window size and the $y$-axis the localization speedup. For each window size speedups were determined using $t$-tests on the localization times for the 30 pairs of data runs. All results are significant at the 95% level. The graph shows that with increasing window size (*i.e.* decreasing processing power), the localization speedup increases. At small window sizes the speedup is 20-50%, but it goes up to 2.7 times for larger windows, demonstrating the benefits of the RTPF approach over traditional particle filters. Ultimately, for very large window sizes, the speedup decreases again, which is due to the fact that none of the approaches is able to reduce the error below 200cm within the run time of an experiment.

# 5 Conclusions

In this paper we tackled the problem of particle filtering under the constraint of limited computing resources. Our approach makes near-optimal use of sensor information by dividing sample sets between all available observations and then representing the state as a mixture of sample sets. Next we optimize the mixing weights in order to be as close to the true posterior distribution as possible. Optimization is performed efficiently by gradient descent using a Monte Carlo approximation of the gradients.

We showed that RTPF produces significant performance improvements in a robot localization task. The results indicate that our approach outperforms all alternative methods for dealing with limited computation. Furthermore, RTPF localized the robot more than 2.7 times faster than the original particle filter approach, which skips sensor data. Based on these results, we expect our method to be highly valuable in a wide range of real-time applications of particle filters. RTPF yields maximal performance gain for data streams containing highly valuable sensor data occuring at unpredictable time points.

The idea of approximating belief states by mixtures has also been used in the context of dynamic Bayesian networks [9]. However, Boyen and Koller use mixtures to represent belief states at a specific point in time, not over multiple time steps. Our work is motivated by real-time constraints that are not present in [9].

So far RTPF uses fixed sample sizes and fixed window sizes. The next natural step is to adapt these two "structural parameters" to further speed up the computation. For example, by the method of [2] we can change the sample size on-the-fly, which in turn allows us to change the window size. Ongoing experiments suggest that this combination yields further performance improvements: When the state uncertainty is high, many samples are used and these samples are spread out over multiple observations. On the other hand, when the uncertainty is low, the number of samples is very small and RTPF becomes identical to the vanilla particle filter with one update (sample set) per observation.

# 6 Acknowledgements

This research is sponsored in part by the National Science Foundation (CAREER grant number 0093406) and by DARPA (MICA program).

## Footnotes

[1]Note that typically the individual predictions $p(x_{t_j} \mid x_{t_{j-1}}, u_{t_{j-1}})$ can be "concatenated" so that only two predictions for each trajectory have to be performed, one before and one after the corresponding observation.

# References

[1] A. Doucet, N. de Freitas, and N. Gordon, editors. *Sequential Monte Carlo in Practice*. Springer-Verlag, New York, 2001.

[2] D. Fox. KLD-sampling: Adaptive particle filters and mobile robot localization. In *Advances in Neural Information Processing Systems (NIPS)*, 2001.

[3] D. Fox, S. Thrun, F. Dellaert, and W. Burgard. Particle filters for mobile robot localization. In Doucet et al. [1].

[4] P. Del Moral and L. Miclo. Branching and interacting particle systems approximations of feynman-kac formulae with applications to non linear filtering. In *Seminaire de Probabilites XXXIV*, number 1729 in Lecture Notes in Mathematics. Springer-Verlag, 2000.

[5] T. M. Cover and J. A. Thomas. *Elements of Information Theory*. Wiley Series in Telecommunications. Wiley, New York, 1991.

[6] W. Poland and R. Shachter. Mixtures of Gaussians and minimum relative entropy techniques for modeling continuous uncertainties. In *Proc. of the Conference on Uncertainty in Artificial Intelligence (UAI)*, 1993.

[7] T. Jaakkola and M. Jordan. Improving the mean field approximation via the use of mixture distributions. In *Learning in Graphical Models*. Kluwer, 1997.

[8] P. R. Cohen. *Empirical methods for artificial intelligence*. MIT Press, 1995.

[9] X. Boyen and D. Koller. Tractable inference for complex stochastic processes. In *Proc. of the Conference on Uncertainty in Artificial Intelligence (UAI)*, 1998.
